# A mathematical model of axon guidance by diffusible factors

**Geoffrey J. Goodhill**
Georgetown Institute for Cognitive and Computational Sciences
Georgetown University Medical Center
3970 Reservoir Road
Washington DC 20007
geoff@giccs.georgetown.edu

## Abstract

In the developing nervous system, gradients of target-derived diffusible factors play an important role in guiding axons to appropriate targets. In this paper, the shape that such a gradient might have is calculated as a function of distance from the target and the time since the start of factor production. Using estimates of the relevant parameter values from the experimental literature, the spatiotemporal domain in which a growth cone could detect such a gradient is derived. For large times, a value for the maximum guidance range of about 1 mm is obtained. This value fits well with experimental data. For smaller times, the analysis predicts that guidance over longer ranges may be possible. This prediction remains to be tested.

## 1  Introduction

In the developing nervous system, growing axons are guided to targets that may be some distance away. Several mechanisms contribute to this (reviewed in Tessier-Lavigne & Goodman (1996)). One such mechanism is the diffusion of a factor from the target through the extracellular space, creating a gradient of increasing concentration that axons can sense and follow. In the central nervous system, such a process seems to occur in at least three cases: the guidance of axons from the trigeminal ganglion to the maxillary process in the mouse (Lumsden & Davies, 1983, 1986), of commissural axons in the spinal cord to the floor plate (Tessier-Lavigne et al., 1988), and of axons and axonal branches from the corticospinal tract to the basilar pons (Heffner et al., 1990). The evidence for this comes from both in vivo and in vitro experiments. For the latter, a piece of target tissue is embedded in a three dimensional collagen gel near to a piece of tissue containing the appropriate

population of neurons. Axon growth is then observed directed towards the target, implicating a target-derived diffusible signal. In vivo, for the systems described, the target is always less than 500 $\mu$m from the population of axons. In vitro, where the distance between axons and target can readily be varied, guidance is generally not seen for distances greater than $500 - 1000$ $\mu$m. Can such a limit be explained in terms of the mathematics of diffusion?

There are two related constraints that the distribution of a diffusible factor must satisfy to provide an effective guidance cue at a point. Firstly, the *absolute concentration* of factor must not be too small or too large. Secondly, the *fractional change in concentration* of factor across the width of the gradient-sensing apparatus, generally assumed to be the growth cone, must be sufficiently large. These constraints are related because in both cases the problem is to overcome statistical noise. At very low concentrations, noise exists due to thermal fluctuations in the number of molecules of factor in the vicinity of the growth cone (analyzed in Berg & Purcell (1977)). At higher concentrations, the limiting source of noise is stochastic variation in the amount of binding of the factor to receptors distributed over the growth cone. At very high concentrations, all receptors will be saturated and no gradient will be apparent. The closer the concentration is to the upper or lower limits, the higher the gradient that is needed to ensure detection (Devreotes & Zigmond, 1988; Tessier-Lavigne & Placzek, 1991). The limitations these constraints impose on the guidance range of a diffusible factor are now investigated. For further discussion see Goodhill (1997; 1998).

## 2   Mathematical model

Consider a source releasing factor with diffusion constant $D$ cm$^2$/sec, at rate $q$ moles/sec, into an infinite, spatially uniform three-dimensional volume. Initially, zero decay of the factor is assumed. For radially symmetric Fickian diffusion in three dimensions, the concentration $C(r, t)$ at distance $r$ from the source at time $t$ is given by

$$C(r,t) = \frac{q}{4\pi Dr}\text{erfc}\frac{r}{\sqrt{4Dt}} \tag{1}$$

(see e.g. Crank (1975)), where erfc is the complementary error function. The percentage change in concentration $p$ across a small distance $\Delta r$ (the width of the growth cone) is given by

$$p = -\frac{\Delta r}{r}\left[1 + \frac{r}{\sqrt{\pi Dt}}\frac{e^{-r^2/4Dt}}{\text{erfc}(r/\sqrt{4Dt})}\right] \tag{2}$$

This function has two perhaps surprising characteristics. Firstly, for fixed $r$, $|p|$ *decreases* with $t$. That is, the largest gradient at any distance occurs immediately after the source starts releasing factor. For large $t$, $|p|$ asymptotes at $\Delta r/r$. Secondly, for fixed $t < \infty$, numerical results show that $p$ is *nonmonotonic* with $r$. In particular it decreases with distance, reaches a minimum, then increases again. The position of this minimum moves to larger distances as $t$ increases.

The general characteristics of the above constraints can be summarized as follows. (1) At small times after the start of production the factor is very unevenly distributed. The concentration $C$ falls quickly to almost zero moving away from the source, the gradient is steep, and the percentage change across the growth cone $p$ is everywhere large. (2) As time proceeds the factor becomes more evenly distributed. $C$ everywhere increases, but $p$ everywhere decreases. (3) For large times, $C$ tends to an inverse variation with the distance from the source $r$, while $|p|$ tends

to $\Delta r/r$ independent of all other parameters. This means that, for large times, the maximum distance over which guidance by diffusible factors is possible scales linearly with growth cone diameter $\Delta r$.

# 3  Parameter values

**Diffusion constant, $D$.** Crick (1970) estimated the diffusion constant in cytoplasm for a molecule of mass 0.3 - 0.5 kDa to be about $10^{-6}$ cm$^2$/sec. Subsequently, a direct determination of the diffusion constant for a molecule of mass 0.17 kDa in the aqueous cytoplasm of mammalian cells yielded a value of about $3.3 \times 10^{-6}$ cm$^2$/sec (Mastro et al., 1984). By fitting a particular solution of the diffusion equation to their data on limb bud determination by gradients of a morphogenetically active retinoid, Eichele & Thaller (1987) calculated a value of $10^{-7}$ cm$^2$/sec for this molecule (mass 348.5 kDa) in embryonic limb tissue. One chemically identified diffusible factor known to be involved in axon guidance is the protein netrin-1, which has a molecular mass of about 75 kDa (Kennedy et al., 1994). $D$ should scale roughly inversely with the radius of a molecule, i.e. with the cube root of its mass. Taking the value of $3.3 \times 10^{-6}$ cm$^2$/sec and scaling it by $(170/75,000)^{1/3}$ yields $4.0 \times 10^{-7}$ cm$^2$/sec. This paper therefore considers $D = 10^{-6}$ cm$^2$/sec and $D = 10^{-7}$ cm$^2$/sec.

**Rate of production of factor $q$.** This is hard to estimate in vivo: some insight can be gained by considering in vitro experiments. Gundersen & Barrett (1979) found a turning response in chick spinal sensory axons towards a nearby pipette filled with a solution of NGF. They estimated the rate of outflow from their pipette to be 1 $\mu$l/hour, and found an effect when the concentration in the pipette was as low as 0.1 nM NGF (Tessier-Lavigne & Placzek, 1991). This corresponds to a $q$ of $3 \times 10^{-11}$ nM/sec. Lohof et al. (1992) studied growth cone turning induced by a gradient of cell-membrane permeant cAMP from a pipette containing a 20 mM solution and a release rate of the order of 0.5 pl/sec: $q = 10^{-5}$ nM/sec. Below a further calculation for $q$ is performed, which suggests an appropriate value may be $q = 10^{-7}$ nM/sec.

**Growth cone diameter, $\Delta r$.** For the three systems mentioned above, the diameter of the main body of the growth cone is less than 10 $\mu$m. However, this ignores filopodia, which can increase the effective width for gradient sensing purposes. The values of 10 $\mu$m and 20 $\mu$m are considered below.

**Minimum concentration for gradient detection.** Studies of leukocyte chemotaxis suggest that when gradient detection is limited by the dynamics of receptor binding rather than physical limits due to a lack of molecules of factor, optimal detection occurs when the concentration at the growth cone is equal to the dissociation constant for the receptor (Zigmond, 1981; Devreotes & Zigmond, 1988). Such studies also suggest that the low concentration limit is about 1% of the dissociation constant (Zigmond, 1981). The transmembrane protein Deleted in Colorectal Cancer (DCC) has recently been shown to possess netrin-1 binding activity, with an order of magnitude estimate for the dissociation constant of 10 nM (Keino-Masu et al, 1996). For comparison, the dissociation constant of the low-affinity NGF receptor P75 is about 1 nM (Meakin & Shooter, 1992). Therefore, low concentration limits of both $10^{-1}$ nM and $10^{-2}$ nM will be considered.

**Maximum concentration for gradient detection.** Theoretical considerations suggest that, for leukocyte chemotaxis, sensitivity to a fixed gradient should fall off symmetrically in a plot against the log of background concentration, with the peak at the dissociation constant for the receptor (Zigmond, 1981). Raising the con-

centration to several hundred times the dissociation constant appears to prevent axon guidance (discussed in Tessier-Lavigne & Placzek (1991)). At concentrations very much greater than the dissociation constant, the number of receptors may be downregulated, reducing sensitivity (Zigmond, 1981). Given the dissociation constants above, 100 nM thus constitutes a reasonable upper bound on concentration.

**Minimum percentage change detectable by a growth cone, $p$.** By establishing gradients of a repellent, membrane-bound factor directly on a substrate and measuring the response of chick retinal axons, Baier & Bonhoeffer (1992) estimated $p$ to be about 1%. Studies of cell chemotaxis in various systems have suggested optimal values of 2%: for concentrations far from the dissociation constant for the receptor, $p$ is expected to be larger (Devreotes & Zigmond, 1988). Both $p = 1\%$ and $p = 2\%$ are considered below.

## 4  Results

In order to estimate bounds for the rate of production of factor $q$ for biological tissue, the empirical observation is used that, for collagen gel assays lasting of the order of one day, guidance is generally seen over distances of at most 500 $\mu$m (Lumsden & Davies, 1983, 1986; Tessier-Lavigne et al., 1988). Assume first that this is constrained by the low concentration limit. Substituting the above parameters (with $D = 10^{-7}\,\text{cm}^2/\text{sec}$) into equation 1 and specifying that $C(500\mu\text{m}, 1\,\text{day}) = 0.01\,\text{nM}$ gives $q \approx 10^{-9}\,\text{nM/sec}$. On the other hand, assuming constraint by the high concentration limit, i.e. $C(500\mu\text{m}, 1\,\text{day}) = 100\,\text{nM}$, gives $q \approx 10^{-5}\,\text{nM/sec}$. Thus it is reasonable to assume that, roughly, $10^{-9}\,\text{nM/sec} < q < 10^{-5}\,\text{nM/sec}$. The results discussed below use a value in between, namely $q = 10^{-7}\,\text{nM/sec}$.

The constraints arising from equations 1 and 2 are plotted in figure 1. The cases of $D = 10^{-6}\,\text{cm}^2/\text{sec}$ and $D = 10^{-7}\,\text{cm}^2/\text{sec}$ are shown in (A,C) and (B,D) respectively. In all four pictures the constraints $C = 0.01\,\text{nM}$ and $C = 0.1\,\text{nM}$ are plotted. In (A,B) the gradient constraint $p = 1\%$ is shown, whereas in (C,D) $p = 2\%$ is shown. These are for a growth cone diameter of 10 $\mu$m. The graph for a 2% change and a growth cone diameter of 20 $\mu$m is identical to that for a 1% change and a diameter of 10 $\mu$m. Each constraint is satisfied for regions to the left of the relevant line. The line $C = 100\,\text{nM}$ is approximately coincident with the vertical axis in all cases. For these parameters, the high concentration limit does not therefore prevent gradient detection until the axons are within a few microns of the source, and it is thus assumed that it is not an important constraint.

As expected, for large $t$ the gradient constraint asymptotes at $\Delta r / r = p$, i.e. $r = 1000\,\mu\text{m}$ for $p = 1\%$ and $r = 500\,\mu\text{m}$ for $p = 2\%$ and a 10 $\mu$m growth cone. That is, the gradient constraint is satisfied at all times when the distance from the source is less than 500 $\mu$m for $p = 2\%$ and $\Delta r = 10\,\mu$m. The gradient constraint lines end to the right because at earlier times $p$ exceeds the critical value over all distances (since the formula for $p$ is non-monotonic with $r$, there is sometimes another branch of each $p$ curve (not shown) off the graph to the right). As $t$ increases from zero, guidance is initially limited only by the concentration constraint. The maximum distance over which guidance can occur increases smoothly with $t$, reaching for instance 1500 $\mu$m (assuming a concentration limit of 0.01 nM) after about 2 hours for $D = 10^{-6}\,\text{cm}^2/\text{sec}$ and about 6 hours for $D = 10^{-7}\,\text{cm}^2/\text{sec}$. However at a particular time, the gradient constraint starts to take effect and rapidly reduces the maximum range of guidance towards the asymptotic value as $t$ increases. This time (for $p = 2\%$) is about 2 hours for $D = 10^{-6}\,\text{cm}^2/\text{sec}$, and about one day for $D = 10^{-7}\,\text{cm}^2/\text{sec}$. It is clear from these pictures that although the exact size of

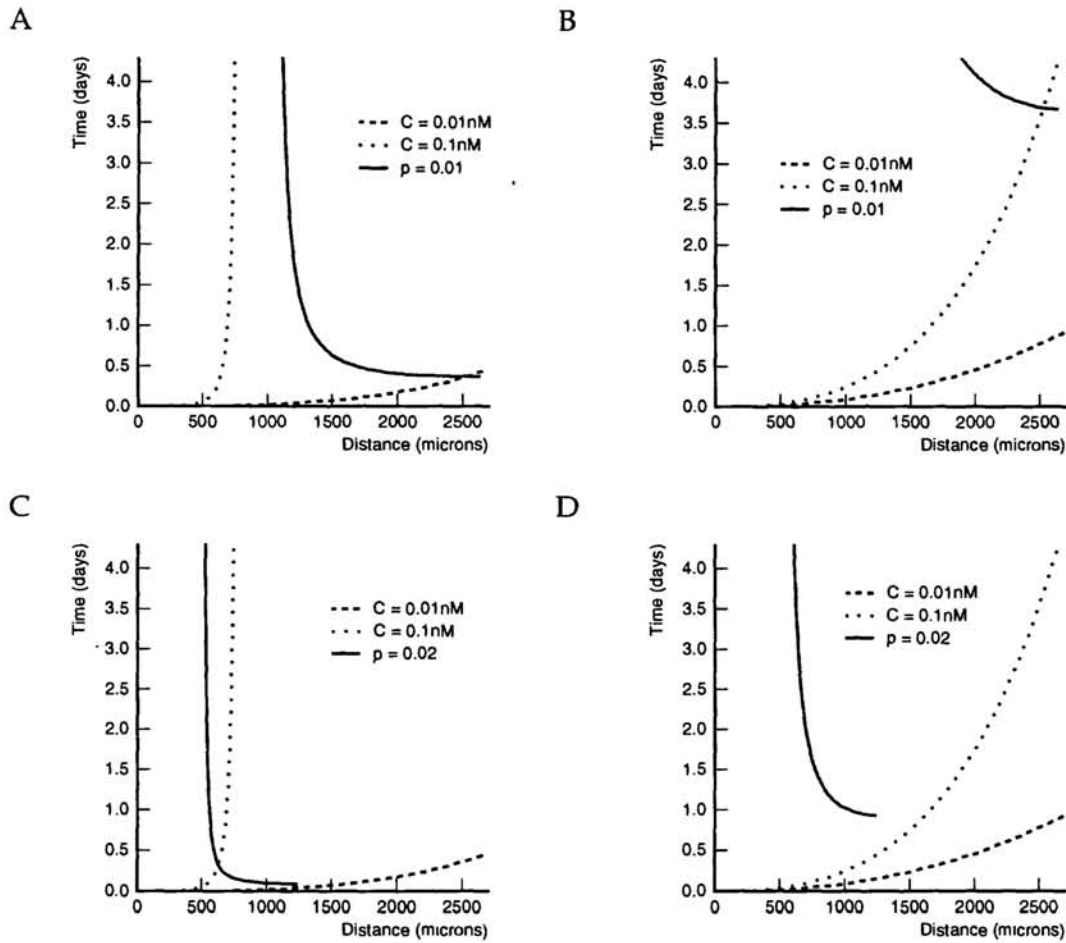

Figure 1: Graphs showing how the gradient constraint (solid line) interacts with the minimum concentration constraint (dashed/dotted lines) to limit guidance range, and how these constraints evolve over time. The top row (A,B) is for $p = 1\%$, the bottom row (C,D) for $p = 2\%$. The left column (A,C) is for $D = 10^{-6}$ cm$^2$/sec, the right column (B,D) for $D = 10^{-7}$ cm$^2$/sec. Each constraint is satisfied to the left of the appropriate curve. It can be seen that for $D = 10^{-6}$cm$^2$/sec the gradient limit quickly becomes the dominant constraint on maximum guidance range. In contrast for $D = 10^{-7}$ cm$^2$/sec, the concentration limit is the dominant constraint at times up to several days. However after this the gradient constraint starts to take effect and rapidly reduces the maximum guidance range.

the diffusion constant does not affect the position of the asymptote for the gradient constraint, it does play an important role in the interplay of constraints while the gradient is evolving. The effect is however subtle: reducing $D$ from $10^{-6}$ cm$^2$/sec to $10^{-7}$ cm$^2$/sec increases the time for the $C = 0.01$ nM limit to reach 2000 $\mu$m, but *decreases* the time for the $C = 0.1$ nM limit to reach 2000 $\mu$m.

## 5 Discussion

Taking the gradient constraint to be a fractional change of at least 2% across a growth cone of width of 10 $\mu$m or 20 $\mu$m yields asymptotic values for the maximum distance over which guidance can occur once the gradient has stabilized of 500 $\mu$m and 1000 $\mu$m respectively. This fits well with both in vitro data, and the fact that for the systems mentioned in the introduction, the growing axons are always less than 500 $\mu$m from the target in vivo. The concentration limits seem to provide a weaker constraint than the gradient limit on the maximum distances possible. However, this is very dependent on the value of $q$, which has only been very roughly estimated: if $q$ is significantly less than $10^{-7}$ nM/sec, the low concentration limits will provide more restrictive constraints ($q$ may well have different values in different target tissues). The gradient constraint curves are independent of $q$. The gradient constraint therefore provides the most robust explanation for the observed guidance limit.

The model makes the prediction that guidance over longer distances than have hitherto been observed may be possible before the gradient has stabilized. In the early stages following the start of factor production the concentration falls off more steeply, providing more effective guidance. The time at which guidance range is a maximum depends on the diffusion constant $D$. For a rapidly diffusing molecule ($D \approx 10^{-6}$cm$^2$/sec) this occurs after only a few hours. For a more slowly diffusing molecule however ($D \approx 10^{-7}$cm$^2$/sec) this occurs after a few days, which would be easier to investigate in vitro. In vivo, molecules such as netrin-1 may thus be large because, during times immediately following the start of production by the source, there could be a definite benefit (i.e. steep gradient) to a slowly-diffusing molecule. Also, it is conceivable that Nature has optimized the start of production of factor relative to the time that guidance is required in order to exploit an evolving gradient for extended range. This could be especially important in larger animals, where axons may need to be guided over longer distances in the developing embryo.

## Bibliography

Baier, H. & Bonhoeffer, F. (1992). Axon guidance by gradients of a target-derived component. *Science*, **255**, 472-475.

Berg, H.C. and Purcell, E.M. (1977). Physics of chemoreception. *Biophysical Journal*, **20**, 193-219.

Crick, F. (1970). Diffusion in embryogenesis. *Nature*, **255**, 420-422.

Crank, J. (1975). The mathematics of diffusion, Second edition. Oxford, Clarendon.

Devreotes, P.N. & Zigmond, S.H. (1988). Chemotaxis in eukaryotic cells: a focus on leukocytes and *Dictyostelium*. *Ann. Rev. Cell. Biol.*, **4**, 649-686.

Eichele, G. & Thaller, C. (1987). Characterization of concentration gradients of a morphogenetically active retinoid in the chick limb bud. *J. Cell. Biol.*, **105**, 1917-1923.

Goodhill, G.J. (1997). Diffusion in axon guidance. *Eur. J. Neurosci.*, **9**, 1414-1421.

Goodhill, G.J. (1998). Mathematical guidance for axons. *Trends. Neurosci.*, in press.

Gundersen, R.W. & Barrett, J.N. (1979). Neuronal chemotaxis: chick dorsal-root axons turn toward high concentrations of nerve growth factor. *Science*, **206**, 1079-1080.

Heffner, C.D., Lumsden, A.G.S. & O'Leary, D.D.M. (1990). Target control of collateral extension and directional growth in the mammalian brain. *Science*, **247**, 217-220.

Keino-Masu, K., Masu, M., Hinck, L., Leonardo, E.D., Chan, S.S.-Y., Culotti, J.G. & Tessier-Lavigne, M. (1996). *Deleted in Colorectal Cancer (DCC)* encodes a netrin receptor. *Cell*, **87**, 175-185.

Kennedy, T.E., Serafini, T., de al Torre, J.R. & Tessier-Lavigne, M. (1994). Netrins are diffusible chemotropic factors for commissural axons in the embryonic spinal cord. *Cell*, **78**, 425-435.

Lohof, A.M., Quillan, M., Dan, Y, & Poo, M-m. (1992). Asymmetric modulation of cytosolic cAMP activity induces growth cone turning. *J. Neurosci.*, **12**, 1253-1261.

Lumsden, A.G.S. & Davies, A.M. (1983). Earliest sensory nerve fibres are guided to peripheral targets by attractants other than nerve growth factor. *Nature*, **306**, 786-788.

Lumsden, A.G.S. & Davies, A.M. (1986). Chemotropic effect of specific target epithelium in the developing mammalian nervous system. *Nature*, **323**, 538-539.

Mastro, A.M., Babich, M.A., Taylor, W.D. & Keith, A.D. (1984). Diffusion of a small molecule in the cytoplasm of mammalian cells. *Proc. Nat. Acad. Sci. USA*, **81**, 3414-3418.

Meakin, S.O. & Shooter, E.M. (1992). The nerve growth family of receptors. *Trends. Neurosci.*, **15**, 323-331.

Tessier-Lavigne, M. & Placzek, M. (1991). Target attraction: are developing axons guided by chemotropism? *Trends Neurosci.*, **14**, 303-310.

Tessier-Lavigne, M. & Goodman, C.S. (1996). The molecular biology of axon guidance. *Science*, **274**, 1123-1133.

Tessier-Lavigne, M., Placzek, M., Lumsden, A.G.S., Dodd, J. & Jessell, T.M. (1988). Chemotropic guidance of developing axons in the mammalian central nervous system. *Nature*, **336**, 775-778.

Tranquillo, R.T. & Lauffenburger, D.A. (1987). Stochastic model of leukocyte chemosensory movement. *J. Math. Biol.*, **25**, 229-262.

Zigmond, S.H. (1981). Consequences of chemotactic peptide receptor modulation for leukocyte orientation. *J. Cell. Biol.*, **88**, 644-647.